# Local Dimensionality Reduction

**Stefan Schaal** [1,2,4]
sschaal@usc.edu
http://www-slab.usc.edu/sschaal

**Sethu Vijayakumar** [3,1]
sethu@cs.titech.ac.jp
http://ogawa-
www.cs.titech.ac.jp/~sethu

**Christopher G. Atkeson** [4]
cga@cc.gatech.edu
http://www.cc.gatech.edu/
fac/Chris.Atkeson

[1]ERATO Kawato Dynamic Brain Project (JST), 2-2 Hikaridai, Seika-cho, Soraku-gun, 619-02 Kyoto
[2]Dept. of Comp. Science & Neuroscience, Univ. of South. California HNB-103, Los Angeles CA 90089-2520
[3]Department of Computer Science, Tokyo Institute of Technology, Meguro-ku, Tokyo-152
[4]College of Computing, Georgia Institute of Technology, 801 Atlantic Drive, Atlanta, GA 30332-0280

## Abstract

If globally high dimensional data has locally only low dimensional distributions, it is advantageous to perform a local dimensionality reduction before further processing the data. In this paper we examine several techniques for local dimensionality reduction in the context of locally weighted linear regression. As possible candidates, we derive local versions of factor analysis regression, principle component regression, principle component regression on joint distributions, and partial least squares regression. After outlining the statistical bases of these methods, we perform Monte Carlo simulations to evaluate their robustness with respect to violations of their statistical assumptions. One surprising outcome is that locally weighted partial least squares regression offers the best average results, thus outperforming even factor analysis, the theoretically most appealing of our candidate techniques.

## 1 INTRODUCTION

Regression tasks involve mapping a $n$-dimensional continuous input vector $\mathbf{x} \in \mathfrak{R}^n$ onto a $m$-dimensional output vector $\mathbf{y} \in \mathfrak{R}^m$. They form a ubiquitous class of problems found in fields including process control, sensorimotor control, coordinate transformations, and various stages of information processing in biological nervous systems. This paper will focus on spatially localized learning techniques, for example, kernel regression with Gaussian weighting functions. Local learning offer advantages for real-time incremental learning problems due to fast convergence, considerable robustness towards problems of negative interference, and large tolerance in model selection (Atkeson, Moore, & Schaal, 1997; Schaal & Atkeson, in press). Local learning is usually based on interpolating data from a local neighborhood around the query point. For high dimensional learning problems, however, it suffers from a bias/variance dilemma, caused by the nonintuitive fact that "… [in high dimensions] if neighborhoods are *local*, then they are almost surely *empty*, whereas if a neighborhood is not *empty*, then it is not *local*." (Scott, 1992, p.198). Global learning methods, such as sigmoidal feedforward networks, do not face this

problem as they do not employ neighborhood relations, although they require strong prior knowledge about the problem at hand in order to be successful.

Assuming that local learning in high dimensions is a hopeless, however, is not necessarily warranted: being *globally* high dimensional does not imply that data remains high dimensional if viewed *locally*. For example, in the control of robot arms and biological arms we have shown that for estimating the inverse dynamics of an arm, a globally 21-dimensional space reduces on average to 4-6 dimensions locally (Vijayakumar & Schaal, 1997). A local learning system that can robustly exploit such locally low dimensional distributions should be able to avoid the curse of dimensionality.

In pursuit of the question of what, in the context of local regression, is the "right" method to perform local dimensionality reduction, this paper will derive and compare several candidate techniques under i) perfectly fulfilled statistical prerequisites (e.g., Gaussian noise, Gaussian input distributions, perfectly linear data), and ii) less perfect conditions (e.g., non-Gaussian distributions, slightly quadratic data, incorrect guess of the dimensionality of the true data distribution). We will focus on nonlinear function approximation with locally weighted linear regression (LWR), as it allows us to adapt a variety of global linear dimensionality reduction techniques, and as LWR has found widespread application in several local learning systems (Atkeson, Moore, & Schaal, 1997; Jordan & Jacobs, 1994; Xu, Jordan, & Hinton, 1996). In particular, we will derive and investigate locally weighted principal component regression (LWPCR), locally weighted joint data principal component analysis (LWPCA), locally weighted factor analysis (LWFA), and locally weighted partial least squares (LWPLS). Section 2 will briefly outline these methods and their theoretical foundations, while Section 3 will empirically evaluate the robustness of these methods using synthetic data sets that increasingly violate some of the statistical assumptions of the techniques.

## 2 METHODS OF DIMENSIONALITY REDUCTION

We assume that our regression data originate from a generating process with two sets of observables, the "inputs" $\tilde{\mathbf{x}}$ and the "outputs" $\tilde{\mathbf{y}}$. The characteristics of the process ensure a functional relation $\tilde{\mathbf{y}} = f(\tilde{\mathbf{x}})$. Both $\tilde{\mathbf{x}}$ and $\tilde{\mathbf{y}}$ are obtained through some measurement device that adds independent mean zero noise of different magnitude in each observable, such that $\mathbf{x} = \tilde{\mathbf{x}} + \varepsilon_x$ and $\mathbf{y} = \mathbf{y} + \varepsilon_y$. For the sake of simplicity, we will only focus on one-dimensional output data ($m=1$) and functions $f$ that are either linear or slightly quadratic, as these cases are the most common in nonlinear function approximation with locally linear models. Locality of the regression is ensured by weighting the error of each data point with a weight from a Gaussian kernel:

$$w_i = \exp\left(-0.5\left(\mathbf{x}_i - \mathbf{x}_q\right)^T \mathbf{D}\left(\mathbf{x}_i - \mathbf{x}_q\right)\right) \tag{1}$$

$\mathbf{x}_q$ denotes the query point, and $\mathbf{D}$ a positive semi-definite distance metric which determines the size and shape of the neighborhood contributing to the regression (Atkeson et al., 1997). The parameters $\mathbf{x}_q$ and $\mathbf{D}$ can be determined in the framework of nonparametric statistics (Schaal & Atkeson, in press) or parametric maximum likelihood estimations (Xu et al, 1995)— for the present study they are determined manually since their origin is secondary to the results of this paper. Without loss of generality, all our data sets will set $\mathbf{x}_q$ to the zero vector, compute the weights, and then translate the input data such that the locally weighted mean, $\bar{\mathbf{x}} = \sum w_i \mathbf{x}_i / \sum w_i$, is zero. The output data is equally translated to be mean zero. Mean zero data is necessary for most of techniques considered below. The (translated) input data is summarized in the rows of the matrix $\mathbf{X}$, the corresponding (translated) outputs are the elements of the vector $\mathbf{y}$, and the corresponding weights are in the diagonal matrix $\mathbf{W}$. In some cases, we need the joint input and output data, denoted as $\mathbf{Z} = [\mathbf{X}\ \mathbf{y}]$.

## 2.1 FACTOR ANALYSIS (LWFA)

Factor analysis (Everitt, 1984) is a technique of dimensionality reduction which is the most appropriate given the generating process of our regression data. It assumes the observed data $\mathbf{z}$ was produced by a mean zero independently distributed $k$-dimensional vector of factors $\mathbf{v}$, transformed by the matrix $\mathbf{U}$, and contaminated by mean zero independent noise $\varepsilon$ with diagonal covariance matrix $\Omega$:

$$\mathbf{z} = \mathbf{U}\mathbf{v} + \varepsilon, \quad \text{where} \quad \mathbf{z} = \left[\mathbf{x}^T, y\right]^T \quad \text{and} \quad \varepsilon = \left[\varepsilon_x^T, \varepsilon_y\right]^T \tag{2}$$

If both $\mathbf{v}$ and $\varepsilon$ are normally distributed, the parameters $\Omega$ and $\mathbf{U}$ can be obtained iteratively by the Expectation-Maximization algorithm (EM) (Rubin & Thayer, 1982). For a linear regression problem, one assumes that $\mathbf{z}$ was generated with $\mathbf{U}=[\mathbf{I}, \beta]^T$ and $\mathbf{v} = \tilde{\mathbf{x}}$, where $\beta$ denotes the vector of regression coefficients of the linear model $y = \beta^I \mathbf{x}$, and $\mathbf{I}$ the identity matrix. After calculating $\Omega$ and $\mathbf{U}$ by EM in joint data space as formulated in (2), an estimate of $\beta$ can be derived from the conditional probability $P(y \mid \mathbf{x})$. As all distributions are assumed to be normal, the expected value of $y$ is the mean of this conditional distribution. The locally weighted version (LWFA) of $\beta$ can be obtained together with an estimate of the factors $\mathbf{v}$ from the joint weighted covariance matrix $\Psi$ of $\mathbf{z}$ and $\mathbf{v}$:

$$E\left\{\begin{bmatrix} y \\ \mathbf{v} \end{bmatrix} \middle| \mathbf{x}\right\} = \begin{bmatrix} \hat{\beta}^T \\ \mathbf{B} \end{bmatrix}\mathbf{x} = \Psi_{21}\Psi_{11}^{-1}\mathbf{x}, \quad \text{where} \quad \Psi = [\mathbf{Z}^T, \mathbf{V}^T]\mathbf{W}\begin{bmatrix} \mathbf{Z} \\ \mathbf{V} \end{bmatrix}\bigg/ \sum w_i = \tag{3}$$

$$= \begin{bmatrix} \Omega + \mathbf{U}\mathbf{U}^T & \mathbf{U} \\ \mathbf{U}^T & \mathbf{I} \end{bmatrix} = \begin{bmatrix} \Psi_{11}(= n \times n) & \Psi_{12}(= n \times (m+k)) \\ \Psi_{21}(= (m+k) \times n) & \Psi_{22}(= (m+k) \times (m+k)) \end{bmatrix}$$

where $E\{\cdot\}$ denotes the expectation operator and $\mathbf{B}$ a matrix of coefficients involved in estimating the factors $\mathbf{v}$. Note that unless the noise $\varepsilon$ is zero, the estimated $\beta$ is different from the true $\beta$ as it tries to average out the noise in the data.

## 2.2 JOINT-SPACE PRINCIPAL COMPONENT ANALYSIS (LWPCA)

An alternative way of determining the parameters $\beta$ in a reduced space employs locally weighted principal component analysis (LWPCA) in the joint data space. By defining the largest $k+1$ principal components of the weighted covariance matrix of $\mathbf{Z}$ as $\mathbf{U}$:

$$\mathbf{U} = \left[eigenvectors\left(\sum w_i (\mathbf{z}_i - \bar{\mathbf{z}})(\mathbf{z}_i - \bar{\mathbf{z}})^T \middle/ \sum w_i\right)\right]_{\max(1:k+1)} \tag{4}$$

and noting that the eigenvectors in $\mathbf{U}$ are unit length, the matrix inversion theorem (Horn & Johnson, 1994) provides a means to derive an efficient estimate of $\beta$

$$\beta = \mathbf{U}_x\left(\mathbf{U}_y^T - \mathbf{U}_y^T\left(\mathbf{U}_y\mathbf{U}_y^T - \mathbf{I}\right)^{-1}\mathbf{U}_y\mathbf{U}_y^T\right), \quad \text{where} \quad \mathbf{U} = \begin{bmatrix} \mathbf{U}_x(= n \times k) \\ \mathbf{U}_y(= m \times k) \end{bmatrix} \tag{5}$$

In our one dimensional output case, $\mathbf{U}_y$ is just a $(1 \times k)$-dimensional row vector and the evaluation of (5) does not require a matrix inversion anymore but rather a division.

If one assumes normal distributions in all variables as in LWFA, LWPCA is the special case of LWFA where the noise covariance $\Omega$ is spherical, i.e., the same magnitude of noise in all observables. Under these circumstances, the subspaces spanned by $\mathbf{U}$ in both methods will be the same. However, the regression coefficients of LWPCA will be different from those of LWFA unless the noise level is zero, as LWFA optimizes the coefficients according to the noise in the data (Equation (3)). Thus, for normal distributions and a correct guess of $k$, LWPCA is always expected to perform worse than LWFA.

## 2.3  PARTIAL LEAST SQUARES (LWPLS, LWPLS_1)

Partial least squares (Wold, 1975; Frank & Friedman, 1993) recursively computes orthogonal projections of the input data and performs single variable regressions along these projections on the residuals of the previous iteration step. A locally weighted version of partial least squares (LWPLS) proceeds as shown in Equation (6) below.

As all single variable regressions are ordinary univariate least-squares minimizations, LWPLS makes the same statistical assumption as ordinary linear regressions, i.e., that only output variables have additive noise, but input variables are noiseless. The choice of the projections **u**, however, introduces an element in LWPLS that remains statistically still debated (Frank & Friedman, 1993), although, interestingly, there exists a strong similarity with the way projections are chosen in Cascade Correlation (Fahlman & Lebiere, 1990). A peculiarity of LWPLS is that it also regresses the inputs of the previous step against the projected inputs **s** in order to ensure the orthogonality of all the projections **u**. Since LWPLS chooses projections in a very powerful way, it can accomplish optimal function fits with only one single projections (i.e.,

| For Training: | For Lookup: |
|---|---|
| Initialize: | Initialize: |
| $\mathbf{D}_0 = \mathbf{X}, \quad \mathbf{e}_0 = \mathbf{y}$ | $\mathbf{d}_0 = \mathbf{x}, \; y = 0$ |
| For i = 1 to k : | For i = 1 to k : |
| $\mathbf{u}_i = \mathbf{D}_{i-1}^T \mathbf{W} \mathbf{e}_{i-1}$ | $s_i = \mathbf{d}_{i-1}^T \mathbf{u}_i$ |
| $\mathbf{s}_i = \mathbf{D}_{i-1} \mathbf{u}_i$ | $y = y + \beta_i s_i$ |
| $\beta_i = \dfrac{\mathbf{s}_i^T \mathbf{W} \mathbf{e}_{i-1}}{\mathbf{s}_i^T \mathbf{W} \mathbf{s}_i}$ | $\mathbf{d}_i = \mathbf{d}_{i-1} - s_i \mathbf{p}_i$ |
| $\mathbf{p}_i = \dfrac{\mathbf{D}_{i-1}^T \mathbf{W} \mathbf{s}_i}{\mathbf{s}_i^T \mathbf{W} \mathbf{s}_i}$ | |
| $\mathbf{D}_i = \mathbf{D}_{i-1} - \mathbf{s}_i \mathbf{p}_i^T$ | (6) |

$k=1$) for certain input distributions. We will address this issue in our empirical evaluations by comparing $k$-step LWPLS with 1-step LWPLS, abbreviated LWPLS_1.

## 2.4  PRINCIPAL COMPONENT REGRESSION (LWPCR)

Although not optimal, a computationally efficient techniques of dimensionality reduction for linear regression is principal component regression (LWPCR) (Massy, 1965). The inputs are projected onto the largest $k$ principal components of the weighted covariance matrix of the input data by the matrix **U**:

$$\mathbf{U} = \left[ eigenvectors\left( \sum w_i \left(\mathbf{x}_i - \bar{\mathbf{x}}\right)\left(\mathbf{x}_i - \bar{\mathbf{x}}\right)^T / \sum w_i \right) \right]_{\max(1:k)} \tag{7}$$

The regression coefficients $\beta$ are thus calculated as:

$$\beta = \left( \mathbf{U}^T \mathbf{X}^T \mathbf{W} \mathbf{X} \mathbf{U} \right)^{-1} \mathbf{U}^T \mathbf{X}^T \mathbf{W} \mathbf{y} \tag{8}$$

Equation (8) is inexpensive to evaluate since after projecting **X** with **U**, $\mathbf{U}^T \mathbf{X}^T \mathbf{W} \mathbf{X} \mathbf{U}$ becomes a diagonal matrix that is easy to invert. LWPCR assumes that the inputs have additive spherical noise, which includes the zero noise case. As during dimensionality reduction LWPCR does not take into account the output data, it is endangered by clipping input dimensions with low variance which nevertheless have important contribution to the regression output. However, from a statistical point of view, it is less likely that low variance inputs have significant contribution in a linear regression, as the confidence bands of the regression coefficients increase inversely proportionally with the variance of the associated input. If the input data has non-spherical noise, LWPCR is prone to focus the regression on irrelevant projections.

## 3  MONTE CARLO EVALUATIONS

In order to evaluate the candidate methods, data sets with 5 inputs and 1 output were randomly generated. Each data set consisted of 2,000 training points and 10,000 test points, distributed either uniformly or nonuniformly in the unit hypercube. The outputs were

generated by either a linear or quadratic function. Afterwards, the 5-dimensional input space was projected into a 10-dimensional space by a randomly chosen distance preserving linear transformation. Finally, Gaussian noise of various magnitudes was added to both the 10-dimensional inputs and one dimensional output. For the test sets, the additive noise in the outputs was omitted. Each regression technique was localized by a Gaussian kernel (Equation (1)) with a 10-dimensional distance metric $\mathbf{D}=10*\mathbf{I}$ ($\mathbf{D}$ was manually chosen to ensure that the Gaussian kernel had sufficiently many data points and no "data holes" in the fringe areas of the kernel) . The precise experimental conditions followed closely those suggested by Frank and Friedman (1993):

- 2 kinds of linear functions $y = \beta_{lin}^T \mathbf{x}$ for:    i) $\beta_{lin} = [1,1,1,1,1]^T$,    ii) $\beta_{lin} = [1,2,3,4,5]^T$

- 2 kinds of quadratic functions $y = \beta_{lin}^T \mathbf{x} + \beta_{quad}^T [x_1^2, x_2^2, x_3^2, x_4^2, x_5^2]^T$ for:

  i) $\beta_{lin} = [1,1,1,1,1]^T$ and $\beta_{quad} = 0.1[1,1,1,1,1]^T$, and ii) $\beta_{lin} = [1,2,3,4,5]^T$ and $\beta_{quad} = 0.1[1,4,9,16,25]^T$

- 3 kinds of noise conditions, each with 2 sub-conditions:
  i) only output noise:    a) low noise:    *local* signal/noise ratio lsnr=20,
              and    b) high noise:    lsnr=2,
  ii) equal noise in inputs and outputs:
          a) low noise $\varepsilon_{x,n} = \varepsilon_y = N(0, 0.01^2)$, $n \in [1,2,...,10]$,
     and    b) high noise $\varepsilon_{x,n} = \varepsilon_y = N(0, 0.1^2)$, $n \in [1,2,...,10]$,
  iii) unequal noise in inputs and outputs:
          a) low noise: $\varepsilon_{x,n} = N(0, (0.01n)^2)$, $n \in [1,2,...,10]$ and lsnr=20,
     and    b) high noise: $\varepsilon_{x,n} = N(0, (0.01n)^2)$, $n \in [1,2,...,10]$ and lsnr=2,

- 2 kinds of input distributions: i) uniform in unit hyper cube, ii) uniform in unit hyper cube excluding data points which activate a Gaussian weighting function (1) at $c = [0.5, 0, 0, 0, 0]^T$ with $\mathbf{D}=10*\mathbf{I}$ more than $w=0.2$ (this forms a "hyper kidney" shaped distribution)

Every algorithm was run[*] 30 times on each of the 48 combinations of the conditions. Additionally, the complete test was repeated for three further conditions varying the dimensionality—called factors in accordance with LWFA—that the algorithms assumed to be the true dimensionality of the 10-dimensional data from $k$=4 to 6, i.e., too few, correct, and too many factors. The average results are summarized in Figure 1.

Figure 1a,b,c show the summary results of the three factor conditions. Besides averaging over the 30 trials per condition, each mean of these charts also averages over the two input distribution conditions and the linear and quadratic function condition, as these four cases are frequently observed violations of the statistical assumptions in nonlinear function approximation with locally linear models. In Figure 1b the number of factors equals the underlying dimensionality of the problem, and all algorithms are essentially performing equally well. For perfectly Gaussian distributions in all random variables (not shown separately), LWFA's assumptions are perfectly fulfilled and it achieves the best results, however, almost indistinguishable closely followed by LWPLS. For the "unequal noise condition", the two PCA based techniques, LWPCA and LWPCR, perform the worst since—as expected—they choose suboptimal projections. However, when violating the statistical assumptions, LWFA loses parts of its advantages, such that the summary results become fairly balanced in Figure 1b.

The quality of function fitting changes significantly when violating the correct number of factors, as illustrated in Figure 1a,c. For too few factors (Figure 1a), LWPCR performs worst because it randomly omits one of the principle components in the input data, without respect to how important it is for the regression. The second worse is LWFA: according to its assumptions it believes that the signal it cannot model must be noise, leading to a degraded estimate of the data's subspace and, consequently, degraded regression results. LWPLS has a clear lead in this test, closely followed by LWPCA and LWPLS_1.

---

[*] Except for LWFA, all methods can evaluate a data set in non-iterative calculations. LWFA was trained with EM for maximally 1000 iterations or until the log-likelihood increased less than 1.e-10 in one iteration.

For too many factors than necessary (Figure 1c), it is now LWPCA which degrades. This effect is due to its extracting one very noise contaminated projection which strongly influences the recovery of the regression parameters in Equation (4). All other algorithms perform almost equally well, with LWFA and LWPLS taking a small lead.

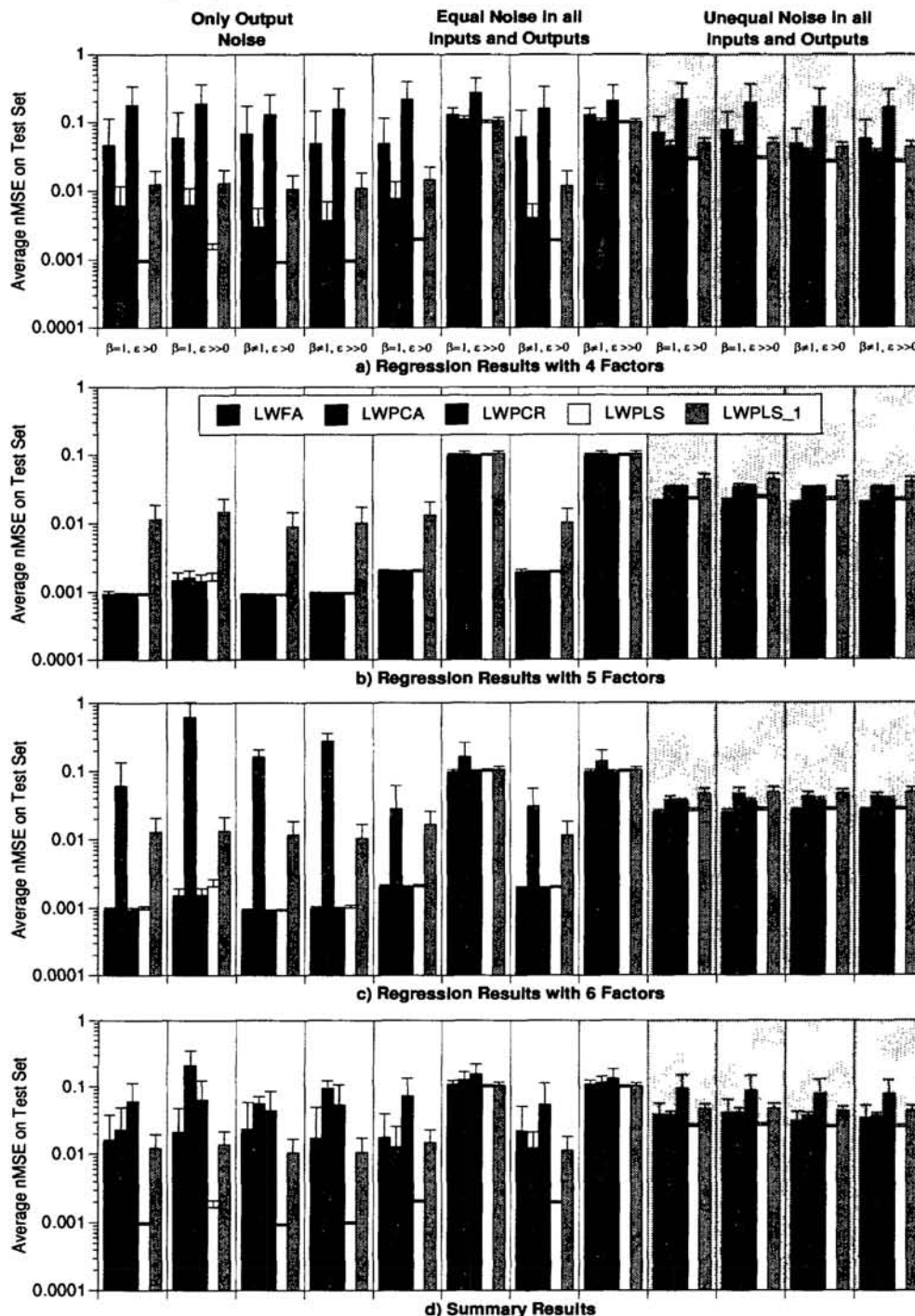

Figure 1: Average summary results of Monte Carlo experiments. Each chart is primarily divided into the three major noise conditions, cf. headers in chart (a). In each noise condition, there are four further subdivision: i) coefficients of linear or quadratic model are equal with low added noise; ii) like i) with high added noise; iii) coefficients of linear or quadratic model are different with low noise added; iv) like iii) with high added noise. Refer to text and descriptions of Monte Carlo studies for further explanations.

# 4 SUMMARY AND CONCLUSIONS

Figure 1d summarizes all the Monte Carlo experiments in a final average plot. Except for LWPLS, every other technique showed at least one clear weakness in one of our "robustness" tests. It was particularly an incorrect number of factors which made these weaknesses apparent. For high-dimensional regression problems, the local dimensionality, i.e., the number of factors, is not a clearly defined number but rather a varying quantity, depending on the way the generating process operates. Usually, this process *does not need* to generate locally low dimensional distributions, however, it often *"chooses"* to do so, for instance, as human arm movements follow stereotypic patterns despite they could generate arbitrary ones. Thus, local dimensionality reduction needs to find *autonomously* the appropriate number of local factor. Locally weighted partial least squares turned out to be a surprisingly robust technique for this purpose, even outperforming the statistically appealing probabilistic factor analysis. As in principal component analysis, LWPLS's number of factors can easily be controlled just based on a variance-cutoff threshold in input space (Frank & Friedman, 1993), while factor analysis usually requires expensive cross-validation techniques. Simple, variance-based control over the number of factors can actually improve the results of LWPCA and LWPCR in practice, since, as shown in Figure 1a, LWPCR is more robust towards overestimating the number of factors, while LWPCA is more robust towards an underestimation. If one is interested in dynamically growing the number of factors while obtaining already good regression results with too few factors, LWPCA and, especially, LWPLS seem to be appropriate—it should be noted how well one factor LWPLS (LWPLS_1) already performed in Figure 1!

In conclusion, since locally weighted partial least squares was equally robust as local weighted factor analysis towards additive noise in both input and output data, and, moreover, superior when mis-guessing the number of factors, it seems to be a most favorable technique for local dimensionality reduction for high dimensional regressions.

## Acknowledgments

The authors are grateful to Geoffrey Hinton for reminding them of partial least squares. This work was supported by the ATR Human Information Processing Research Laboratories. S. Schaal's support includes the German Research Association, the Alexander von Humboldt Foundation, and the German Scholarship Foundation. S. Vijayakumar was supported by the Japanese Ministry of Education, Science, and Culture (Monbusho). C. G. Atkeson acknowledges the Air Force Office of Scientific Research grant F49-6209410362 and a National Science Foundation Presidential Young Investigators Award.

## References

Atkeson, C. G., Moore, A. W., & Schaal, S, (1997a). "Locally weighted learning." *Artificial Intelligence Review*, 11, 1-5, pp.11-73.

Atkeson, C. G., Moore, A. W., & Schaal, S, (1997c). "Locally weighted learning for control." *Artificial Intelligence Review*, 11, 1-5, pp.75-113.

Belsley, D. A., Kuh, E., & Welsch, R. E, (1980). *Regression diagnostics: Identifying influential data and sources of collinearity*. New York: Wiley.

Everitt, B. S, (1984). *An introduction to latent variable models*. London: Chapman and Hall.

Fahlman, S. E. , Lebiere, C, (1990). "The cascade-correlation learning architecture." In: Touretzky, D. S. (Ed.), *Advances in Neural Information Processing Systems II*, pp.524-532. Morgan Kaufmann.

Frank, I. E., & Friedman, J. H, (1993). "A statistical view of some chemometric regression tools." *Technometrics*, 35, 2, pp.109-135.

Geman, S., Bienenstock, E., & Doursat, R, (1992). "Neural networks and the bias/variance dilemma." *Neural Computation*, 4, pp.1-58.

Horn, R. A., & Johnson, C. R, (1994). *Matrix analysis*. Press Syndicate of the University of Cambridge.

Jordan, M. I., & Jacobs, R, (1994). "Hierarchical mixtures of experts and the EM algorithm." *Neural Computation*, 6, 2, pp.181-214.

Massy, W. F, (1965). "Principle component regression in exploratory statistical research." *Journal of the American Statistical Association*, 60, pp.234-246.

Rubin, D. B., & Thayer, D. T, (1982). "EM algorithms for ML factor analysis." *Psychometrika*, 47, 1, 69-76.

Schaal, S., & Atkeson, C. G, (in press). "Constructive incremental learning from only local information." *Neural Computation*.

Scott, D. W, (1992). *Multivariate Density Estimation*. New York: Wiley.

Vijayakumar, S., & Schaal, S, (1997). "Local dimensionality reduction for locally weighted learning." In: *International Conference on Computational Intelligence in Robotics and Automation*, pp.220-225, Monteray, CA, July 10-11, 1997.

Wold, H. (1975). "Soft modeling by latent variables: the nonlinear iterative partial least squares approach." In: Gani, J. (Ed.), *Perspectives in Probability and Statistics, Papers in Honour of M. S. Bartlett*. Acad. Press.

Xu, L., Jordan, M. I., & Hinton, G. E, (1995). "An alternative model for mixture of experts." In: Tesauro, G., Touretzky, D. S., & Leen, T. K. (Eds.), *Advances in Neural Information Processing Systems 7*, pp.633-640. Cambridge, MA: MIT Press.